# Domain Adaptation with Multiple Sources

**Yishay Mansour**
Google Research and
Tel Aviv Univ.
mansour@tau.ac.il

**Mehryar Mohri**
Courant Institute and
Google Research
mohri@cims.nyu.edu

**Afshin Rostamizadeh**
Courant Institute
New York University
rostami@cs.nyu.edu

## Abstract

This paper presents a theoretical analysis of the problem of domain adaptation with multiple sources. For each source domain, the distribution over the input points as well as a hypothesis with error at most $\epsilon$ are given. The problem consists of combining these hypotheses to derive a hypothesis with small error with respect to the target domain. We present several theoretical results relating to this problem. In particular, we prove that standard convex combinations of the source hypotheses may in fact perform very poorly and that, instead, combinations weighted by the source distributions benefit from favorable theoretical guarantees. Our main result shows that, remarkably, for any fixed target function, there exists a distribution weighted combining rule that has a loss of at most $\epsilon$ with respect to *any* target mixture of the source distributions. We further generalize the setting from a single target function to multiple consistent target functions and show the existence of a combining rule with error at most $3\epsilon$. Finally, we report empirical results for a multiple source adaptation problem with a real-world dataset.

## 1  Introduction

A common assumption in theoretical models of learning such as the standard PAC model [16], as well as in the design of learning algorithms, is that training instances are drawn according to the same distribution as the unseen test examples. In practice, however, there are many cases where this assumption does not hold. There can be no hope for generalization, of course, when the training and test distributions vastly differ, but when they are less dissimilar, learning can be more successful.

A typical situation is that of *domain adaptation* where little or no labeled data is at one's disposal for the *target domain*, but large amounts of labeled data from a *source domain* somewhat similar to the target, or hypotheses derived from that source, are available instead. This problem arises in a variety of applications in natural language processing [4, 7, 10], speech processing [8, 9, 11, 13–15], computer vision [12], and many other areas.

This paper studies the problem of domain adaptation with multiple sources, which has also received considerable attention in many areas such as natural language processing and speech processing. An example is the problem of *sentiment analysis* which consists of classifying a text sample such as a movie review, restaurant rating, or discussion boards, or other web pages. Information about a relatively small number of domains such as *movies* or *books* may be available, but little or none can be found for more difficult domains such as *travel*.

We will consider the following problem of multiple source adaptation. For each source $i \in [1, k]$, the learner receives the distribution $D_i$ of the input points corresponding to that source as well as a hypothesis $h_i$ with loss at most $\epsilon$ on that source. The learner's task consists of combining the $k$ hypotheses $h_i$, $i \in [1, k]$, to derive a hypothesis $h$ with small loss with respect to the target distribution. The target distribution is assumed to be a mixture of the distributions $D_i$. We will discuss both the case where the mixture is known to the learner and the case where it is unknown.

Note that the distribution $D_i$ is defined over the input points and bears no information about the labels. In practice, $D_i$ is estimated from large amounts of unlabeled points typically available from source $i$.

An alternative set-up for domain adaptation with multiple sources is one where the learner is not supplied with a good hypothesis $h_i$ for each source but where instead he has access to the labeled training data for each source domain. A natural solution consists then of combining the raw labeled data from each source domain to form a new sample more representative of the target distribution and use that to train a learning algorithm. This set-up and the type of solutions just described have been in fact explored extensively in applications [8, 9, 11, 13–15]. However, several empirical observations motivated our study of hypothesis combination, in addition to the theoretical simplicity and clarity of this framework.

First, in some applications such as very large-vocabulary speech recognition, often the original raw data used to derive each domain-dependent model is no more available [2, 9]. This is because such models are typically obtained as a result of training based on many hours of speech with files occupying hundreds of gigabytes of disk space, while the models derived require orders of magnitude less space. Thus, combining raw labeled data sets is not possible in such cases. Secondly, a combined data set can be substantially larger than each domain-specific data set, which can significantly increase the computational cost of training and make it prohibitive for some algorithms. Thirdly, combining labeled data sets requires the mixture parameters of the target distribution to be known, but it is not clear how to produce a hypothesis with a low error rate with respect to *any* mixture distribution.

Few theoretical studies have been devoted to the problem of adaptation with multiple sources. Ben-David et al. [1] gave bounds for single source adaptation, then Blitzer et al. [3] extended the work to give a bound on the error rate of a hypothesis derived from a weighted combination of the source data sets for the specific case of empirical risk minimization. Crammer et al. [5, 6] also addressed a problem where multiple sources are present but the nature of the problem differs from adaptation since the distribution of the input points is the same for all these sources, only the labels change due to varying amounts of noise. We are not aware of a prior theoretical study of the problem of adaptation with multiple sources analyzed here.

We present several theoretical results relating to this problem. We examine two types of hypothesis combination. The first type is simply based on convex combinations of the $k$ hypotheses $h_i$. We show that this natural and widely used hypothesis combination may in fact perform very poorly in our setting. Namely, we give a simple example of two distributions and two matching hypotheses, each with zero error for their respective distribution, but such that any convex combination has expected absolute loss of $1/2$ for the equal mixture of the distributions. This points out a potentially significant weakness of a convex combination.

The second type of hypothesis combination, which is the main one we will study in this work, takes into account the probabilities derived from the distributions. Namely, the weight of hypothesis $h_i$ on an input $x$ is proportional to $\lambda_i D_i(x)$, were $\lambda$ is the set of mixture weights. We will refer to this method as the *distribution weighted hypothesis combination*. Our main result shows that, remarkably, for any fixed target function, there exists a distribution weighted combining rule that has a loss of at most $\epsilon$ with respect to *any* mixture of the $k$ distributions. We also show that there exists a distribution weighted combining rule that has loss at most $3\epsilon$ with respect to any consistent target function (one for which each $h_i$ has loss $\epsilon$ on $D_i$) and any mixture of the $k$ distributions. In some sense, our results establish that the distribution weighted hypothesis combination is the "right" combination rule, and that it also benefits from a well-founded theoretical guarantee.

The remainder of this paper is organized as follows. Section 2 introduces our theoretical model for multiple source adaptation. In Section 3, we analyze the abstract case where the mixture parameters of the target distribution are known and show that the distribution weighted hypothesis combination that uses as weights these mixture coefficients achieves a loss of at most $\epsilon$. In Section 4, we give a simple method to produce an error of $\Theta(k\epsilon)$ that does not require the prior knowledge of the mixture parameters of the target distribution. Our main results showing the existence of a combined hypothesis performing well regardless of the target mixture are given in Section 5 for the case of a fixed target function, and in Section 6 for the case of multiple target functions. Section 7 reports empirical results for a multiple source adaptation problem with a real-world dataset.

## 2  Problem Set-Up

Let $\mathcal{X}$ be the input space, $f\colon \mathcal{X} \to \mathbb{R}$ the target function to learn, and $L\colon \mathbb{R} \times \mathbb{R} \to \mathbb{R}$ a loss function penalizing errors with respect to $f$. The loss of a hypothesis $h$ with respect to a distribution $D$ and loss function $L$ is denoted by $\mathcal{L}(D, h, f)$ and defined as $\mathcal{L}(D, h, f) = \mathrm{E}_{x \sim D}[L(h(x), f(x))] = \sum_{x \in \mathcal{X}} L(h(x), f(x)) D(x)$. We will denote by $\Delta$ the simplex $\Delta = \{\lambda \colon \lambda_i \geq 0 \wedge \sum_{i=1}^{k} \lambda_i = 1\}$ of $\mathbb{R}^k$.

We consider an adaptation problem with $k$ source domains and a single target domain. The input to the problem is the set of $k$ source distributions $D_1, \ldots, D_k$ and $k$ corresponding hypotheses $h_1, \ldots, h_k$ such that for all $i \in [1, k]$, $\mathcal{L}(D_i, h_i, f) \leq \epsilon$, for a fixed $\epsilon \geq 0$. The distribution of the target domain, $D_T$, is assumed to be a mixture of the $k$ source distributions $D_i$s, that is $D_T(x) = \sum_{i=1}^{k} \lambda_i D_i(x)$, for some unknown mixture weight vector $\lambda \in \Delta$. The adaptation problem consists of combing the hypotheses $h_i$s to derive a hypothesis with small loss on the target domain. Since the target distribution $D_T$ is assumed to be a mixture, we will refer to this problem as the *mixture adaptation problem*.

A *combining rule* for the hypotheses takes as an input the $h_i$s and outputs a single hypothesis $h\colon \mathcal{X} \to \mathbb{R}$. We define two combining rules of particular interest for our purpose: the *linear combining rule* which is based on a parameter $z \in \Delta$ and which sets the hypothesis to $h(x) = \sum_{i=1}^{k} z_i h_i(x)$; and the *distribution weighted combining rule* also based on a parameter $z \in \Delta$ which sets the hypothesis to $h(x) = \sum_{i=1}^{k} \frac{z_i D_i(x)}{\sum_{j=1}^{k} z_j D_j(x)} h_i(x)$ when $\sum_{j=1}^{k} z_j D_j(x) > 0$. This last condition always holds if $D_i(x) > 0$ for all $x \in \mathcal{X}$ and some $i \in [1, k]$. We define $\mathcal{H}$ to be the set of all distribution weighted combining rules. Given the input to the adaptation problem we have implicit information about the target function $f$. We define the set of *consistent target functions*, $\mathcal{F}$, as follows,

$$\mathcal{F} = \{g \colon \forall i \in [1, k], \ \mathcal{L}(D_i, h_i, g) \leq \epsilon\}.$$

By definition, the target function $f$ is an element of $\mathcal{F}$.

We will assume that the following properties hold for the loss function $L$: (i) $L$ is non-negative: $L(x, y) \geq 0$ for all $x, y \in \mathbb{R}$; (ii) $L$ is convex with respect to the first argument: $L(\sum_{i=1}^{k} \lambda_i x_i, y) \leq \sum_{i=1}^{k} \lambda_i L(x_i, y)$ for all $x_1, \ldots, x_k, y \in \mathbb{R}$ and $\lambda \in \Delta$; (iii) $L$ is bounded: there exists $M \geq 0$ such that $L(x, y) \leq M$ for all $x, y \in \mathbb{R}$; (iv) $L(x, y)$ is continuous in both $x$ and $y$; and (v) $L$ is symmetric $L(x, y) = L(y, x)$. The absolute loss defined by $L(x, y) = |x - y|$ will serve as our primary motivating example.

## 3  Known Target Mixture Distribution

In this section we assume that the parameters of the target mixture distribution are known. Thus, the learning algorithm is given $\lambda \in \Delta$ such that $D_T(x) = \sum_{i=1}^{k} \lambda_i D_i(x)$. A good starting point would be to study the performance of a linear combining rule. Namely the classifier $h(x) = \sum_{i=1}^{k} \lambda_i h_i(x)$. While this seems like a very natural classifier, the following example highlights the problematic aspects of this approach.

Consider a discrete domain $\mathcal{X} = \{a, b\}$ and two distributions, $D_a$ and $D_b$, such that $D_a(a) = 1$ and $D_b(b) = 1$. Namely, each distribution puts all the weight on a single element in $\mathcal{X}$. Consider the target function $f$, where $f(a) = 1$ and $f(b) = 0$, and let the loss be the absolute loss. Let $h_0 = 0$ be the function that outputs 0 for all $x \in \mathcal{X}$ and similarly $h_1 = 1$. The hypotheses $h_1$ and $h_0$ have *zero* expected absolute loss on the distributions $D_a$ and $D_b$, respectively, i.e., $\epsilon = 0$. Now consider the target distribution $D_T$ with $\lambda_a = \lambda_b = 1/2$, thus $D_T(a) = D_T(b) = 1/2$. The hypothesis $h(x) = (1/2)h_1(x) + (1/2)h_0(x)$ always outputs $1/2$, and has an absolute loss of $1/2$. Furthermore, for any other parameter $z$ of the linear combining rule, the expected absolute loss of $h(x) = zh_1(x) + (1-z)h_0(x)$ with respect to $D_T$ is exactly $1/2$. We have established the following theorem.

**Theorem 1.** *There is a mixture adaptation problem with $\epsilon = 0$ for which any linear combination rule has expected absolute loss of $1/2$.*

Next we show that the distribution weighted combining rule produces a hypothesis with a low expected loss. Given a mixture $D_T(x) = \sum_{i=1}^{k} \lambda_i D_i(x)$, we consider the distribution weighted combining rule with parameter $\lambda$, which we denote by $h_\lambda$. Recall that,

$$h_\lambda(x) = \sum_{i=1}^{k} \frac{\lambda_i D_i(x)}{\sum_{j=1}^{k} \lambda_j D_j(x)} h_i(x) = \sum_{i=1}^{k} \frac{\lambda_i D_i(x)}{D_T(x)} h_i(x) \ .$$

Using the convexity of $L$ with respect to the first argument, the loss of $h_\lambda$ with respect to $D_T$ and a target $f \in \mathcal{F}$ can be bounded as follows,

$$\mathcal{L}(D_T, h_\lambda, f) = \sum_{x \in X} L(h_\lambda(x), f(x)) D_T(x) \le \sum_{x \in X} \sum_{i=1}^{k} \lambda_i D_i(x) L(h_i(x), f(x)) = \sum_{i=1}^{k} \lambda_i \epsilon_i \le \epsilon,$$

where $\epsilon_i := \mathcal{L}(D_i, h_i, f) \le \epsilon$. Thus, we have derived the following theorem.

**Theorem 2.** *For any mixture adaptation problem with target distribution $D_\lambda(x) = \sum_{i=1}^{k} \lambda_i D_i(x)$, the expected loss of the hypothesis $h_\lambda$ is at most $\epsilon$ with respect to any target function $f \in \mathcal{F}$: $\mathcal{L}(D_\lambda, h_\lambda, f) \le \epsilon$.*

## 4 Simple Adaptation Algorithms

In this section we show how to construct a simple distribution weighted hypothesis that has an expected loss guarantee with respect to any mixture. Our hypothesis $h_u$ is simply based on equal weights, i.e., $u_i = 1/k$, for all $i \in [1, k]$. Thus,

$$h_u(x) = \sum_{i=1}^{k} \frac{(1/k) D_i(x)}{\sum_{j=1}^{k} (1/k) D_j(x)} h_i(x) = \sum_{i=1}^{k} \frac{D_i(x)}{\sum_{j=1}^{k} D_j(x)} h_i(x).$$

We show for $h_u$ an expected loss bound of $k\epsilon$, with respect to any mixture distribution $D_T$ and target function $f \in \mathcal{F}$. (Proof omitted.)

**Theorem 3.** *For any mixture adaptation problem the expected loss of $h_u$ is at most $k\epsilon$, for any mixture distribution $D_T$ and target function $f \in \mathcal{F}$, i.e., $\mathcal{L}(D_T, h_u, f) \le k\epsilon$.*

Unfortunately, the hypothesis $h_u$ can have an expected absolute loss as large as $\Omega(k\epsilon)$. (Proof omitted.)

**Theorem 4.** *There is a mixture adaptation problem for which the expected absolute loss of $h_u$ is $\Omega(k\epsilon)$. Also, for $k = 2$ there is an input to the mixture adaptation problem for which the expected absolute loss of $h_u$ is $2\epsilon - \epsilon^2$.*

## 5 Existence of a Good Hypothesis

In this section, we will show that for any target function $f \in \mathcal{F}$ there is a distribution weighted combining rule $h_z$ that has a loss of at most $\epsilon$ with respect to any mixture $D_T$. We will construct the proof in two parts. In the first part, we will show, using a simple reduction to a zero-sum game, that one can obtain a mixture of $h_z$s that guarantees a loss bounded by $\epsilon$. In the second part, which is the more interesting scenario, we will show that for any target function $f \in \mathcal{F}$ there is a single distribution weighted combining rule $h_z$ that has loss of at most $\epsilon$ with respect to *any* mixture $D_T$. This later part will require the use of Brouwer fixed point theorem to show the existence of such an $h_z$.

### 5.1 Zero-sum game

The adaptation problem can be viewed as a zero-sum game between two players, NATURE and LEARNER. Let the input to the mixture adaptation problem be $D_1, \ldots, D_k$, $h_1, \ldots, h_k$ and $\epsilon$, and fix a target function $f \in \mathcal{F}$. The player NATURE picks a distribution $D_i$ while the player LEARNER selects a distribution weighted combining rule $h_z \in \mathcal{H}$. The loss when NATURE plays $D_i$ and LEARNER plays $h_z$ is $\mathcal{L}(D_i, h_z, f)$. Let us emphasize that the target function $f \in \mathcal{F}$ is fixed beforehand. The objective of NATURE is to maximize the loss and the objective of LEARNER is to minimize the loss. We start with the following lemma,

**Lemma 1.** *Given any mixed strategy of* NATURE, *i.e., a distribution $\mu$ over $D_i$'s, then the following action of* LEARNER $h_\mu \in \mathcal{H}$ *has expected loss at most $\epsilon$, i.e., $\mathcal{L}(D_\mu, h_\mu, f) \leq \epsilon$.*

The proof is identical to that of Theorem 2. This almost establishes that the value of the game is at most $\epsilon$. The technical part that we need to take care of is the fact that the action space of LEARNER is infinite. However, by an appropriate discretization of $\mathcal{H}$ we can derive the following theorem.

**Theorem 5.** *For any target function $f \in \mathcal{F}$ and any $\delta > 0$, there exists a function $h(x) = \sum_{j=1}^m \alpha_j h_{z_j}(x)$, where $h_{z_i} \in \mathcal{H}$, such that $\mathcal{L}(D_T, h, f) \leq \epsilon + \delta$ for any mixture distribution $D_T(x) = \sum_{i=1}^k \lambda_i D_i(x)$.*

Since we can fix $\delta > 0$ to be arbitrarily small, this implies that a linear mixture of distribution weighted combining rules can guarantee a loss of almost $\epsilon$ with respect to any product distribution.

## 5.2 Single distribution weighted combining rule

In the previous subsection, we showed that a mixture of hypotheses in $\mathcal{H}$ would guarantee a loss of at most $\epsilon$. Here, we will considerably strengthen the result and show that there is a *single* hypothesis in $\mathcal{H}$ for which this guarantee holds. Unfortunately our loss is not convex with respect to $h \in \mathcal{H}$, so we need to resort to a more powerful technique, namely the Brouwer fixed point theorem.

For the proof we will need that the distribution weighted combining rule $h_z$ be continuous in the parameter $z$. In general, this does hold due to the existence of points $x \in X$ for which $\sum_{j=1}^k z_j D_j(x) = 0$. To avoid this discontinuity, we will modify the definition of $h_z$ to $h_z^\eta$, as follows.

**Claim 1.** *Let $U$ denote the uniform distribution over $\mathcal{X}$, then for any $\eta > 0$ and $z \in \Delta$, let $h_z^\eta \colon \mathcal{X} \to \mathbb{R}$ be the function defined by*

$$h_z^\eta(x) = \sum_{i=1}^k \frac{z_i D_i(x) + \eta U(x)/k}{\sum_{j=1}^k z_j D_j(x) + \eta U(x)} h_i(x).$$

*Then, for any distribution $D$, $\mathcal{L}(D, h_z^\eta, f)$ is continuous in $z$.*[1]

Let us first state Brouwer's fixed point theorem.

**Theorem 6** (Brouwer Fixed Point Theorem). *For any compact and convex non-empty set $A \subset \mathbb{R}^n$ and any continuous function $f \colon A \to A$, there is a point $x \in A$ such that $f(x) = x$.*

We first show that there exists a distribution weighted combining rule $h_z^\eta$ for which the losses $\mathcal{L}(D_i, h_z^\eta, f)$ are all nearly the same.

**Lemma 2.** *For any target function $f \in \mathcal{F}$ and any $\eta, \eta' > 0$, there exists $z \in \Delta$, with $z_i \neq 0$ for all $i \in [1, k]$, such that the following holds for the distribution weighted combining rule $h_z^\eta \in \mathcal{H}$:*

$$\mathcal{L}(D_i, h_z^\eta, f) = \gamma + \eta' - \frac{\eta'}{z_i k} \leq \gamma + \eta'$$

*for any $1 \leq i \leq k$, where $\gamma = \sum_{j=1}^k z_j \mathcal{L}(D_j, h_z^\eta, f)$.*

*Proof.* Fix $\eta' > 0$ and let $\mathcal{L}_i^z = \mathcal{L}(D_i, h_z^\eta, f)$ for all $z \in \Delta$ and $i \in [1, m]$. Consider the mapping $\phi \colon \Delta \to \Delta$ defined for all $z \in \Delta$ by $[\phi(z)]_i = (z_i \mathcal{L}_i^z + \eta'/k) / (\sum_{j=1}^k z_j \mathcal{L}_j^z + \eta')$, where $[\phi(z)]_i$, is the $i$th coordinate of $\phi(x)$, $i \in [1, m]$. By Claim 1, $\phi$ is continuous. Thus, by Brouwer's Fixed Point Theorem, there exists $z \in \Delta$ such that $\phi(z) = z$. This implies that $z_i = (z_i \mathcal{L}_i^z + \eta'/k) / (\sum_{j=1}^k z_j \mathcal{L}_j^z + \eta')$. Since $\eta' > 0$, we must have $z_i \neq 0$ for any $i \in [1, m]$. Thus, we can divide by $z_i$ and write $\mathcal{L}_i^z + \eta'/(z_i k) = (\sum_{j=1}^k z_j \mathcal{L}_j^z) + \eta'$. Therefore, $\mathcal{L}_i^z = \gamma + \eta' - \eta'/(z_i k)$ with $\gamma = \sum_{j=1}^k z_j \mathcal{L}_j^z$. $\qquad\square$

Note that the lemma just presented does not use the structure of the distribution weighted combining rule, but only the fact that the loss is continuous in the parameter $z \in \Delta$. The lemma applies as well to the linear combination rule and provides the same guarantee. The real crux of the argument is, as shown in the next lemma, that $\gamma$ is small for a distribution weighted combining rule (while it can be very large for a linear combination rule).

**Lemma 3.** *For any target function $f \in \mathcal{F}$ and any $\eta, \eta' > 0$, there exists $z \in \Delta$ such that $\mathcal{L}(D_\lambda, h_z^\eta, f) \leq \epsilon + \eta M + \eta'$ for any $\lambda \in \Delta$.*

*Proof.* Let $z$ be the parameter guaranteed in Lemma 2. Then $\mathcal{L}(D_i, h_z^\eta, f) = \gamma + \eta' - \eta'/(z_i k) \leq \gamma + \eta'$, for $1 \leq i \leq k$. Consider the mixture $D_z$, i.e., set the mixture parameter to be $z$. Consider the quantity $\mathcal{L}(D_z, h_z^\eta, f)$. On the one hand, by definition, $\mathcal{L}(D_z, h_z^\eta, f) = \sum_{i=1}^k z_i \mathcal{L}(D_i, h_z^\eta, f)$ and thus $\mathcal{L}(D_z, h_z^\eta, f) = \gamma$. On the other hand,

$$
\begin{aligned}
\mathcal{L}(D_z, & h_z^\eta, f) \\
&= \sum_{x \in \mathcal{X}} D_z(x) L(h_z^\eta(x), f(x)) \leq \sum_{x \in \mathcal{X}} \frac{D_z(x)}{D_z(x) + \eta U(x)} \left( \sum_{i=1}^k (z_i D_i(x) + \frac{\eta U(x)}{k}) L(h_i(x), f(x)) \right) \\
&\leq \sum_{x \in \mathcal{X}} \left( \sum_{i=1}^k z_i D_i(x) L(h_i(x), f(x)) \right) + \sum_{x \in \mathcal{X}} \eta M U(x) \\
&= \sum_{i=1}^k z_i \mathcal{L}(D_i, h_i, f) + \eta M = \sum_{i=1}^k z_i \epsilon_i + \eta M \leq \epsilon + \eta M \ .
\end{aligned}
$$

Therefore $\gamma \leq \epsilon + \eta M$. To complete the proof, note that the following inequality holds for any mixture $D_\lambda$:

$$
\mathcal{L}(D_\lambda, h_z^\eta, f) = \sum_{i=1}^k \lambda_i \mathcal{L}(D_i, h_z^\eta, f) \leq \gamma + \eta',
$$

which is at most $\epsilon + \eta M + \eta'$. $\qquad\square$

By setting $\eta = \delta/(2M)$ and $\eta' = \delta/2$, we can derive the following theorem.

**Theorem 7.** *For any target function $f \in \mathcal{F}$ and any $\delta > 0$, there exists $\eta > 0$ and $z \in \Delta$, such that $\mathcal{L}(D_\lambda, h_z^\eta, f) \leq \epsilon + \delta$ for any mixture parameter $\lambda$.*

## 6 Arbitrary target function

The results of the previous section show that for any *fixed* target function there is a good distribution weighted combining rule. In this section, we wish to extend these results to the case where the target function is not fixed in advanced. Thus, we seek a single distribution weighted combining rule that can perform well for *any* $f \in \mathcal{F}$ and *any* mixture $D_\lambda$. Unfortunately, we are not able to prove a bound of $\epsilon + o(\epsilon)$ but only a bound of $3\epsilon$. To show this bound we will show that for any $f_1, f_2 \in \mathcal{F}$ and *any* hypothesis $h$ the difference of loss is bounded by at most $2\epsilon$.

**Lemma 4.** *Assume that the loss function $L$ obeys the triangle inequality, i.e., $L(f, h) \leq L(f, g) + L(g, h)$. Then for any $f, f' \in \mathcal{F}$ and any mixture $D_T$, the inequality $\mathcal{L}(D_T, h, f') \leq \mathcal{L}(D_T, h, f) + 2\epsilon$ holds for any hypothesis $h$.*

*Proof.* Since our loss function obeys the triangle inequality, for any functions $f, g, h$, the following holds, $\mathcal{L}(D, f, h) \leq \mathcal{L}(D, f, g) + \mathcal{L}(D, g, h)$. In our case, we observe that replacing $g$ with any $f' \in F$ gives, $\mathcal{L}(D_\lambda, f, h) \leq \mathcal{L}(D_\lambda, f', h) + \mathcal{L}(D_\lambda, f, f')$. We can bound the term $\mathcal{L}(D_\lambda, f, f')$ with a similar inequality, $\mathcal{L}(D_\lambda, f, f') \leq \mathcal{L}(D_\lambda, f, h_\lambda) + \mathcal{L}(D_\lambda, f', h_\lambda) \leq 2\epsilon$, where $h_\lambda$ is the distribution weighted combining rule produced by choosing $z = \lambda$ and using Theorem 2. Therefore, for any $f, f' \in F$ we have, $\mathcal{L}(D_\lambda, f, h) \leq \mathcal{L}(D_\lambda, f', h) + 2\epsilon$, which completes the proof. $\qquad\square$

We derived the following corollary to Theorem 7.

**Corollary 1.** *Assume that the loss function $L$ obeys the triangle inequality. Then, for any $\delta > 0$, there exists $\eta > 0$ and $z \in \Delta$, such that for any mixture parameter $\lambda$ and any $f \in \mathcal{F}$, $\mathcal{L}(D_\lambda, h_z^\eta, f) \leq 3\epsilon + \delta$.*

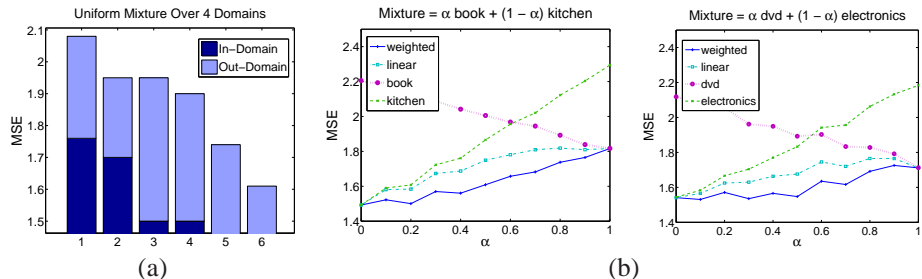

Figure 1: (a) MSE performance for a target mixture of four domains (1: books, 2: dvd, 3: electronics, 4: kitchen 5: linear, 6: weighted). (b) MSE performance under various mixtures of two source domains, plot left: `book` and `kitchen`, plot right: `dvd` and `electronics`.

## 7   Empirical results

This section reports the results of our experiments with a distribution weighted combining rule using real-world data. In our experiments, we fixed a mixture target distribution $D_\lambda$ and considered the distribution weighted combining rule $h_z$, with $z = \lambda$. Since we used real-world data, we did not have access to the domain distributions. Instead, we modeled each distribution and used large amounts of unlabeled data available for each source to estimate the model's parameters. One could have thus expected potentially significantly worse empirical results than the theoretical ones, but this turned out not to be an issue in our experiments.

We used the sentiment analysis dataset found in [4].[2]  The data consists of review text and rating labels, taken from `amazon.com` product reviews within four different categories (domains). These four domains consist of `book`, `dvd`, `electronics` and `kitchen` reviews, where each domain contains 2000 data points. [3] In our experiments, we fixed a mixture target distribution $D_\lambda$ and considered the distribution weighted combining rule $h_z$, with $z = \lambda$.

In our first experiment, we considered mixtures of all four domains, where the test set was a uniform mixture of 600 points, that is the union of 150 points taken uniformly at random from each domain. The remaining 1,850 points from each domain were used to train the base hypotheses.[4]  We compared our proposed weighted combining rule to the linear combining rule. The results are shown in Figure 1(a). They show that the base hypotheses perform poorly on the mixture test set, which justifies the need for adaptation. Furthermore, the distribution weighted combining rule is shown to perform at least as well as the worst in-domain performance of a base hypothesis, as expected from our bounds. Finally, we observe that this real-world data experiment gives an example in which a linear combining rule performs poorly compared to the distribution weighted combining rule.

In other experiments, we considered the mixture of two domains, where the mixture is varied according to the parameter $\alpha \in \{0.1, 0.2, \ldots, 1.0\}$. For each plot in Figure 1 (b), the test set consists of $600\alpha$ points from the first domain and $600(1 - \alpha)$ points from the second domain, where the first and second domains are made clear in the figure. The remaining points that were not used for testing were used to train the base hypotheses. The results show the linear shift from one domain to the other, as is evident from the performance of the two base hypotheses. The distribution weighted combining rule outperforms the base hypotheses as well as the linear combining rule.

Thus, our preliminary experiments suggest that the distribution weighted combining rule performs well in practice and clearly outperforms a simple linear combining rule. Furthermore, using statistical language models as approximations to the distribution oracles seem to be sufficient in practice and can help produce a good distribution weighted combining rule.

## 8  Conclusion

We presented a theoretical analysis of the problem of adaptation with multiple sources. Domain adaptation is an important problem that arises in a variety of modern applications where limited or no labeled data is available for a target application and our analysis can be relevant in a variety of situations. The theoretical guarantees proven for the distribution weight combining rule provide it with a strong foundation. Its empirical performance with a real-world data set further motivates its use in applications. Much of the results presented were based on the assumption that the target distribution is some mixture of the source distributions. A further analysis suggests however that our main results can be extended to arbitrary target distributions.

**Acknowledgments**

We thank Jennifer Wortman for helpful comments on an earlier draft of this paper and Ryan McDonald for discussions and pointers to data sets. The work of M. Mohri and A. Rostamizadeh was partly supported by the New York State Office of Science Technology and Academic Research (NYSTAR).

## Footnotes

[1] In addition to continuity, the perturbation to $h_z$, $h_z^\eta$, also helps us ensure that none of the mixture weights $z_i$ is zero in the proof of the Lemma 2 .

[2]http://www.seas.upenn.edu/~mdredze/datasets/sentiment/.

[3]The rating label, an integer between 1 and 5, was used as a regression label, and the loss measured by the mean squared error (MSE). All base hypotheses were generated using Support Vector Regression (SVR) [17] with the trade-off parameters $C = 8, \epsilon = 0.1$, and a Gaussian kernel with parameter $g = 0.00078$. The SVR solutions were obtained using the libSVM software library ( http://www.csie.ntu.edu.tw/~cjlin/libsvm/). Our features were defined as the set of unigrams appearing five times or more in all domains. This defined about 4000 unigrams. We used a binary feature vector encoding the presence or absence of these frequent unigrams to define our instances. To model the domain distributions, we used a unigram statistical language model trained on the same corpus as the one used to define the features. The language model was created using the GRM library (http://www.research.att.com/~fsmtools/grm/).

[4]Each experiment was repeated 20 times with random folds. The standard deviation found was far below what could be legibly displayed in the figures.

## References

[1] Shai Ben-David, John Blitzer, Koby Crammer, and Fernando Pereira. Analysis of representations for domain adaptation. In *Proceedings of NIPS 2006*. MIT Press, 2007.

[2] Jacob Benesty, M. Mohan Sondhi, and Yiteng Huang, editors. *Springer Handbook of Speech Processing*. Springer, 2008.

[3] John Blitzer, Koby Crammer, A. Kulesza, Fernando Pereira, and Jennifer Wortman. Learning bounds for domain adaptation. In *Proceedings of NIPS 2007*. MIT Press, 2008.

[4] John Blitzer, Mark Dredze, and Fernando Pereira. Biographies, Bollywood, Boom-boxes and Blenders: Domain Adaptation for Sentiment Classification. In *ACL 2007*, Prague, Czech Republic, 2007.

[5] Koby Crammer, Michael Kearns, and Jennifer Wortman. Learning from Data of Variable Quality. In *Proceedings of NIPS 2005*, 2006.

[6] Koby Crammer, Michael Kearns, and Jennifer Wortman. Learning from multiple sources. In *Proceedings of NIPS 2006*, 2007.

[7] Mark Dredze, John Blitzer, Pratha Pratim Talukdar, Kuzman Ganchev, Joao Graca, and Fernando Pereira. Frustratingly Hard Domain Adaptation for Parsing. In *CoNLL 2007*, Prague, Czech Republic, 2007.

[8] Jean-Luc Gauvain and Chin-Hui. Maximum a posteriori estimation for multivariate gaussian mixture observations of markov chains. *IEEE Transactions on Speech and Audio Processing*, 2(2):291–298, 1994.

[9] Frederick Jelinek. *Statistical Methods for Speech Recognition*. The MIT Press, 1998.

[10] Jing Jiang and ChengXiang Zhai. Instance Weighting for Domain Adaptation in NLP. In *Proceedings of ACL 2007*, pages 264–271, Prague, Czech Republic, 2007. Association for Computational Linguistics.

[11] C. J. Legetter and Phil C. Woodland. Maximum likelihood linear regression for speaker adaptation of continuous density hidden markov models. *Computer Speech and Language*, pages 171–185, 1995.

[12] Aleix M. Martínez. Recognizing imprecisely localized, partially occluded, and expression variant faces from a single sample per class. *IEEE Trans. Pattern Anal. Mach. Intell.*, 24(6):748–763, 2002.

[13] S. Della Pietra, V. Della Pietra, R. L. Mercer, and S. Roukos. Adaptive language modeling using minimum discriminant estimation. In *HLT '91: Proceedings of the workshop on Speech and Natural Language*, pages 103–106, Morristown, NJ, USA, 1992. Association for Computational Linguistics.

[14] Brian Roark and Michiel Bacchiani. Supervised and unsupervised PCFG adaptation to novel domains. In *Proceedings of HLT-NAACL*, 2003.

[15] Roni Rosenfeld. A Maximum Entropy Approach to Adaptive Statistical Language Modeling. *Computer Speech and Language*, 10:187–228, 1996.

[16] Leslie G. Valiant. *A theory of the learnable*. ACM Press New York, NY, USA, 1984.

[17] Vladimir N. Vapnik. *Statistical Learning Theory*. Wiley-Interscience, New York, 1998.

